# Softening Discrete Relaxation

**Andrew M. Finch, Richard C. Wilson and Edwin R. Hancock**
Department of Computer Science,
University of York, York, Y01 5DD, UK

## Abstract

This paper describes a new framework for relational graph matching. The starting point is a recently reported Bayesian consistency measure which gauges structural differences using Hamming distance. The main contributions of the work are threefold. Firstly, we demonstrate how the discrete components of the cost function can be softened. The second contribution is to show how the softened cost function can be used to locate matches using continuous non-linear optimisation. Finally, we show how the resulting graph matching algorithm relates to the standard quadratic assignment problem.

## 1 Introduction

Graph matching [6, 5, 7, 2, 3, 12, 11] is a topic of central importance in pattern perception. The main computational issues are how to compare inexact relational descriptions [7] and how to search efficiently for the best match [8]. These two issues have recently stimulated interest in the connectionist literature [9, 6, 5, 10]. For instance, Simic [9], Suganathan *et al.* [10] and Gold *et al.* [6, 5] have addressed the issue of how to expressively measure relational distance. Both Gold and Rangarajan [6] and Suganathan *et al* [10] have shown how non-linear optimisation techniques such as mean-field annealing [10] and graduated assignment [6] can be applied to find optimal matches.

In a recent series of papers we have developed a Bayesian framework for relational graph matching [2, 3, 11, 12]. The novelty resides in the fact that relational consistency is gauged by a probability distribution that uses Hamming distance to measure structural differences between the graphs under match. This new framework has not only been used to match complex infra-red [3] and radar imagery [11], it has also been used to successfully control a graph-edit process [12] of the sort originally proposed by Sanfeliu and Fu [7]. The optimisation of this relational consistency measure has hitherto been confined to the use of discrete update procedures [11, 2, 3]. Examples include discrete relaxation [7, 11], simulated annealing

[4, 3] and genetic search [2]. Our aim in this paper is to consider how the optimisation of the relational consistency measure can be realised by continuous means [6, 10]. Specifically we consider how the matching process can be effected using a non-linear technique similar to mean-field annealing [10] or graduated assignment [6]. In order to achieve this goal we must transform our discrete cost function [11] into a form suitable for optimisation by continuous techniques. The key idea is to exploit the apparatus of statistical physics [13] to compute the effective Gibbs potentials for our discrete relaxation process. The potentials are in-fact weighted sums of Hamming distance enumerated over the consistent relations of the model graph. The quantities of interest in the optimisation process are the derivatives of the global energy function computed from the Gibbs potentials. In the case of our weighted sum of Hamming distance, these derivatives take on a particularly interesting form which provides an intuitive insight into the dynamics of the update process. An experimental evaluation of the technique reveals not only that it is successful in matching noise corrupted graphs, but that it significantly outperforms the optimisation of the standard quadratic energy function.

## 2 Relational Consistency

Our overall goal in this paper is to formulate a non-linear optimisation technique for matching relational graphs. We use the notation $G = (V, E)$ to denote the graphs under match, where $V$ is the set of nodes and $E$ is the set of edges. Our aim in matching is to associate nodes in a graph $G_D = (V_D, E_D)$ representing data to be matched against those in a graph $G_M = (V_M, E_M)$ representing an available relational model. Formally, the matching is represented by a function $f : V_D \to V_M$ from the nodes in the data graph $G_D$ to those in the model graph $G_M$. We represent the structure of the two graphs using a pair of connection matrices. The connection matrix for the data graph consists of the binary array

$$D_{ab} = \begin{cases} 1 & \text{if } (a, b) \in E_D \\ 0 & \text{otherwise} \end{cases} \tag{1}$$

while that for the model graph is

$$M_{\alpha\beta} = \begin{cases} 1 & \text{if } (\alpha, \beta) \in E_M \\ 0 & \text{otherwise} \end{cases} \tag{2}$$

The current state of match between the two graphs is represented by the function $f : V_D \to V_M$. In others words the statement $f(a) = \alpha$ means that the node $a \in V_D$ is matched to the node $\alpha \in V_M$. The binary representation of the current state of match is captured by a set of assignment variables which convey the following meaning

$$s_{a\alpha} = \begin{cases} 1 & \text{if } f(a) = \alpha \\ 0 & \text{otherwise} \end{cases} \tag{3}$$

The basic goal of the matching process is to optimise a consistency-measure which gauges the structural similarity of the matched data graph and the model graph. In a recent series of papers, Wilson and Hancock [11, 12] have shown how consistency of match can be modelled using a Bayesian framework. The basic idea is to construct a probability distribution which models the effect of memoryless matching errors in generating departures from consistency between the data and model graphs. Suppose that $S_\alpha = \alpha \cup \{\beta | (\alpha, \beta) \in E_M\}$ represents the set of nodes that form the immediate contextual neighbourhood of the node $\alpha$ in the model graph.

Further suppose that $\Gamma_a = f(a) \cup \{f(b)|(a,b) \in E_D\}$ represents the set of matches assigned to the contextual neighbourhood of the node $a \in V_D$ of the data graph. Basic to Wilson and Hancock's modelling of relational consistency is to regard the complete set of model-graph relations as mutually exclusive causes from which the potentially corrupt matched model-graph relations arise. As a result, the probability of the matched configuration $\Gamma_a$ can be expressed as a mixture distribution over the corresponding space of model-graph configurations

$$P(\Gamma_a) = \sum_{\alpha \in V_M} P(\Gamma_a|S_\alpha)P(S_\alpha) \tag{4}$$

The modelling of the match confusion probabilities $P(\Gamma_a|S_\alpha)$ draws on the assumption that the error process is independent of location. This allows $P(\Gamma_a|S_\alpha)$ to be factorised over its component matches. Individual label errors are further assumed to act with a memoryless probability $P_e$. With these ingredients the probability of the matched neighbourhood $\Gamma_a$ reduces to [11, 12]

$$P(\Gamma_a) = \frac{K_a}{|V_M|} \sum_{\alpha \in V_M} \exp[-\mu H(a, \alpha)] \tag{5}$$

where $K_a = (1 - P_e)^{|\Gamma_a|}$ and the exponential constant is related to the probability of label errors, i.e. $\mu = \ln \frac{(1-P_e)}{P_e}$. Consistency of match is gauged by the "Hamming distance", $H(a, \alpha)$ between the matched relation $\Gamma_a$ and the set of consistent neighbourhood structures $S_\alpha, \forall \alpha \in V_M$ from the model graph. According to our binary representation of the matching process, the distance measure is computed using the connectivity matrices and the assignment variables in the following manner

$$H(a, \alpha) = \sum_{b \in V_D} \sum_{\beta \in V_M} M_{\alpha\beta} D_{ab}(1 - s_{b\beta}) \tag{6}$$

The probability distribution $P(\Gamma_a)$ may be regarded as providing a natural way of modelling departures from consistency at the neighbourhood level. Matching consistency is graded by Hamming distance and controlled hardening may be induced by reducing the label-error probability $P_e$ towards zero.

## 3  The Effective Potential for Discrete Relaxation

We commence the development of our graduated assignment approach to discrete relaxation by computing an effective Gibbs potential $U(\Gamma_a)$ for the matching configuration $\Gamma_a$. In other words, we aim to replace the compound exponential probability distribution appearing in equation (5) by the single Gibbs distribution

$$Q(\Gamma_a) = \frac{\exp\left[-\mu U(\Gamma_a)\right]}{\sum_{\Upsilon} \exp\left[-\mu U(\Upsilon)\right]} \tag{7}$$

Our route to the effective potential is provided by statistical physics. If we represent $P(\Gamma_a)$ by an equivalent Gibbs distribution with an identical partition function, then the equilibrium configurational potential is related to the partial derivative of the log-probability with respect to the coupling constant $\mu$ in the following manner [13]

$$U(\Gamma_a) = -\frac{\partial \ln P(\Gamma_a)}{\partial \mu} \tag{8}$$

Upon substituting for $P(\Gamma_a)$ from equation (5)

$$U(\Gamma_a) = \frac{\sum\limits_{\alpha \in V_M} H(a,\alpha) \exp[-\mu H(a,\alpha)]}{\sum\limits_{\alpha \in V_M} \exp[-\mu H(a,\alpha)]} \tag{9}$$

In other words the neighbourhood Gibbs potentials are simply weighted sums of Hamming distance between the data and model graphs. In fact the local clique potentials display an interesting barrier property. The potential is concentrated at Hamming distance $H \simeq \frac{1}{\mu}$. Both very large and very small Hamming distances contribute insignificantly to the energy function, i.e. $\lim_{H \to 0} H \exp[-\mu H] = 0$ and $\lim_{H \to \infty} H \exp[-\mu H] = 0$.

With the neighbourhood matching potentials to hand, we construct a global "matching-energy" $\mathcal{E} = \sum_{a \in V_D} U(\Gamma_a)$ by summing the contributions over the nodes of the data graph.

## 4 Optimising the Global Cost Function

We are now in a position to develop a continuous update algorithm by softening the discrete ingredients of our graph matching potential. The idea is to compute the derivatives of the global energy given in equation (10) and to effect the softening process using the soft-max idea of Bridle [1].

### 4.1 Softassign

The energy function represented by equations (9) and (10) is defined over the discrete matching variables $s_{a\alpha}$. The basic idea underpinning this paper is to realise a continuous process for updating the assignment variables. The optimal step-size is determined by computing the partial derivatives of the global matching energy with respect to the assignment variables. We commence by computing the derivatives of the contributing neighbourhood Gibbs potentials, i.e.

$$\frac{\partial U(\Gamma_a)}{\partial s_{b\beta}} = \sum_{\alpha \in V_M} \left[ 1 - \mu \left( H(a,\alpha) - U(\Gamma_a) \right) \right] \xi_{a\alpha} \frac{\partial H(a,\alpha)}{\partial s_{b\beta}} \tag{10}$$

where

$$\xi_{a\alpha} = \frac{\exp[-\mu H(a,\alpha)]}{\sum_{\alpha' \in V_M} \exp[-\mu H(a,\alpha')]} \tag{11}$$

To further develop this result, we must compute the derivatives of the Hamming distances. From equation (6) it follows that

$$\frac{\partial H(a,\alpha)}{\partial s_{b\beta}} = -M_{\alpha\beta} D_{ab} \tag{12}$$

It is now a straightforward matter to show that the derivative of the global matching energy is equal to

$$\frac{\partial \mathcal{E}}{\partial s_{b\beta}} = -\sum_{a \in V_D} \sum_{\alpha \in V_M} D_{ab} M_{\alpha\beta} \left[ 1 - \mu \left( H(a,\alpha) - U(\Gamma_a) \right) \right] \xi_{a\alpha} \tag{13}$$

We would like our continuous matching variables to remain constrained to lie within the range $[0, 1]$. Rather than using a linear update rule, we exploit Bridle's soft-max ansatz [1]. In doing this we arrive at an update process which has many features in common with the well-known mean-field equations of statistical physics

$$s_{a\alpha} \leftarrow \frac{\exp\left[-\frac{1}{T}\frac{\partial \mathcal{E}}{\partial s_{a\alpha}}\right]}{\sum\limits_{\alpha' \in V_M} \exp\left[-\frac{1}{T}\frac{\partial \mathcal{E}}{\partial s_{a\alpha'}}\right]} \tag{14}$$

The mathematical structure of this update process is important and deserves further comment. The quantity $\xi_{a\alpha}$ defined in equation (11) naturally plays the role of a matching probability. The first term appearing under the square bracket in equation (13) can therefore be thought of as analogous to the optimal update direction for the standard quadratic cost function [10, 6]; we will discus this relationship in more detail in Section 4.2. The second term modifies this principal update direction by taking into account the weighted fluctuations in the Hamming distance about the effective potential or average Hamming distance. If the average fluctuation is zero, then there is no net modification to the update direction. When the net fluctuation is non-zero, the direction of update is modified so as to compensate for the movement of the mean-value of the effective potential. This corrective tracking process provides an explicit mechanism for maintaining contact with the minimum of the effective potential under rescaling effects induced by changes in the value of the coupling constant $\mu$. Moreover, since the fluctuation term is itself proportional to $\mu$, this has an insignificant effect for $P_e \simeq \frac{1}{2}$ but dominates the update process when $P_e \to 0$.

## 4.2 Quadratic Assignment Problem

Before we proceed to experiment with the new graph matching process, it is interesting to briefly review the standard quadratic formulation of the matching problem investigated by Simic [9], Suganathan et al [10] and Gold and Rangarajan [6]. The common feature of these algorithms is to commence from the quadratic cost function

$$\mathcal{E}_H = -\frac{1}{2} \sum_{a \in V_D} \sum_{\alpha \in V_M} \sum_{b \in V_D} \sum_{\beta \in V_M} D_{ab} M_{\alpha\beta} s_{a\alpha} s_{b\beta} \tag{15}$$

In this case the derivative of the global cost function is linear in the assignment variables, i.e.

$$\frac{\partial \mathcal{E}_H}{\partial s_{b\beta}} = -\frac{1}{2} \sum_{a \in V_D} \sum_{\alpha \in V_M} D_{ab} M_{\alpha\beta} s_{a\alpha} \tag{16}$$

This step size is equivalent to that appearing in equation (14) provided that $\mu = 0$, i.e. $P_e \to \frac{1}{2}$. The update is realised by applying the soft-max ansatz of equation (14). In the next section, we will provide some experimental comparison with the resulting matching process. However, it is important to stress that the update process adopted here is very simplistic and leaves considerable scope for further refinement. For instance, Gold and Rangarajan [6] have exploited the doubly stochastic properties of Sinckhorn matrices to ensure two-way symmetry in the matching process.

## 5   Experiments and Conclusions

Our main aim in this Section is to compare the non-linear update equations with the optimisation of the quadratic matching criterion described in Section 4.2. The data for our study is provided by synthetic Delaunay graphs. These graphs are constructed by generating random dot patterns. Each random dot is used to seed a Voronoi cell. The Delaunay triangulation is the region adjacency graph for the Voronoi cells. In order to pose demanding tests of our matching technique, we have added controlled amounts of corruption to the synthetic graphs. This is effected by deleting and adding a specified fraction of the dots from the initial random patterns. The associated Delaunay graph is therefore subject to structural corruption. We measure the degree of corruption by the fraction of surviving nodes in the corrupted Delaunay graph.

Our experimental protocol has been as follows. For a series of different corruption levels, we have generated a sample of 100 random graphs. The graphs contain 50 nodes each. According to the specified corruption level, we have both added and deleted a predefined fraction of nodes at random locations in the initial graphs so as to maintain their overall size. For each graph we measure the quality of match by computing the fraction of the surviving nodes for which the assignment variables indicate the correct match. The value of the temperature $T$ in the update process has been controlled using a logarithmic annealing schedule of the form suggested by Geman and Geman [4]. We initialise the assignment variables uniformly across the set of matches by setting $s_{a\alpha} = \frac{1}{V_M}$, $\forall a, \alpha$.

We have compared the results obtained with two different versions of the matching algorithm. The first of these involves updating the softened assignment variables by applying the non-linear update equation given in (14). The second matching algorithm involves applying the same optimisation apparatus to the quadratic cost function defined in equation (15) in a simplified form of the quadratic assignment algorithm [6, 10].

Figure 1 shows the final fraction of correct matches for both algorithms. The data curves show the correct matching fraction averaged over the graph samples as a function of the corruption fraction. The main conclusions that can be drawn from these plots is that the new matching technique described in this paper significantly outperforms its conventional quadratic counterpart described in Section 4.2. The main difference between the two techniques resides in the fact that our new method relies on updating with derivatives of the energy function that are non-linear in the assignment variables.

To conclude, our main contribution in this paper has been to demonstrate how the discrete Bayesian relational consistency measure of Wilson and Hancock [11] can be cast in a form that is amenable to continuous non-linear optimisation. We have shown how the method relates to the standard quadratic assignment algorithm extensively studied in the connectionist literature [6, 9, 10]. Moreover, an experimental analysis reveals that the method offers superior performance in terms of noise control.

## References

[1] Bridle J.S. "Training stochastic model recognition algorithms can lead to maximum mutual information estimation of parameters" *NIPS2*, pp. 211-217, 1990.

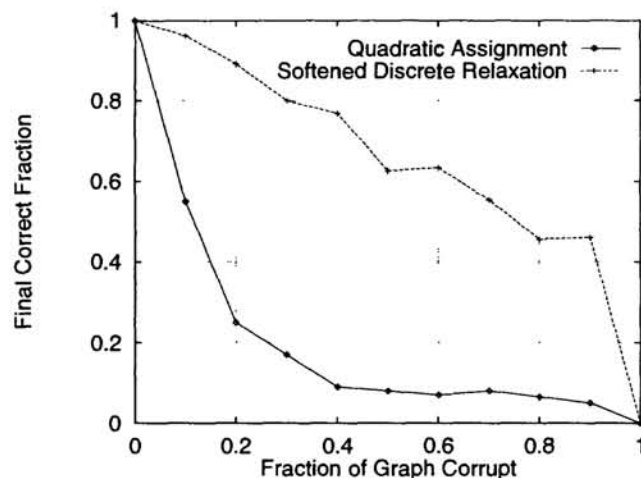

Figure 1: Experimental comparison: softened discrete relaxation (dotted curve); matching using the quadratic cost function (solid curve).

[2] Cross A.D.J., R.C.Wilson and E.R. Hancock, "Genetic search for structural matching", *Proceedings ECCV96*, **LNCS 1064**, pp. 514–525, 1996.

[3] Cross A.D.J. and E.R.Hancock, "Relational matching with stochastic optimisation" *IEEE Computer Society International Symposium on Computer Vision*, pp. 365–370, 1995.

[4] Geman S. and D. Geman, "Stochastic relaxation, Gibbs distributions and Bayesian restoration of images," *IEEE PAMI*, **PAMI-6** , pp.721–741, 1984.

[5] Gold S., A. Rangarajan and E. Mjolsness, "Learning with pre-knowledge: Clustering with point and graph-matching distance measures", *Neural Computation*, **8**, pp. 787–804, 1996.

[6] Gold S. and A. Rangarajan, "A graduated assignment algorithm for graph matching", *IEEE PAMI*, **18**, pp. 377–388, 1996.

[7] Sanfeliu A. and Fu K.S., "A distance measure between attributed relational graphs for pattern recognition", *IEEE SMC*, **13**, pp 353–362, 1983.

[8] Shapiro L. and R.M.Haralick, "Structural description and inexact matching", *IEEE PAMI*, **3**, pp 504–519, 1981.

[9] Simic P., "Constrained nets for graph matching and other quadratic assignment problems", *Neural Computation*, **3** , pp. 268–281, 1991.

[10] Suganathan P.N., E.K. Teoh and D.P. Mital, "Pattern recognition by graph matching using Potts MFT networks", *Pattern Recognition*, **28**, pp. 997–1009, 1995.

[11] Wilson R.C., Evans A.N. and Hancock E.R., "Relational matching by discrete relaxation", *Image and Vision Computing*, **13**, pp. 411–421, 1995.

[12] Wilson R.C and Hancock E.R., "Relational matching with dynamic graph structures", *Proceedings of the Fifth International Conference on Computer Vision*, pp. 450–456, 1995.

[13] Yuille A., "Generalised deformable models, statistical physics and matching problems", *Neural Computation*, **2**, pp. 1-24, 1990.
